# Combined discriminative and generative articulated pose and non-rigid shape estimation

**Leonid Sigal**    **Alexandru Balan**    **Michael J. Black**
Department of Computer Science
Brown University
Providence, RI 02912
{ls, alb, black}@cs.brown.edu

## Abstract

Estimation of three-dimensional articulated human pose and motion from images is a central problem in computer vision. Much of the previous work has been limited by the use of crude generative models of humans represented as articulated collections of simple parts such as cylinders. Automatic initialization of such models has proved difficult and most approaches assume that the size and shape of the body parts are known *a priori*. In this paper we propose a method for automatically recovering a detailed parametric model of non-rigid body shape and pose from monocular imagery. Specifically, we represent the body using a parameterized triangulated mesh model that is learned from a database of human range scans. We demonstrate a discriminative method to directly recover the model parameters from monocular images using a conditional mixture of kernel regressors. This predicted pose and shape are used to initialize a generative model for more detailed pose and shape estimation. The resulting approach allows fully automatic pose and shape recovery from monocular and multi-camera imagery. Experimental results show that our method is capable of robustly recovering articulated pose, shape and biometric measurements (*e.g.* height, weight, *etc.*) in both calibrated and uncalibrated camera environments.

## 1   Introduction

We address the problem of marker-less articulated pose and shape estimation of the human body from images using a detailed parametric body model [3]. Most prior work on marker-less pose estimation and tracking has concentrated on the use of generative Baysian methods [8, 15] that exploit crude models of body shape (*e.g.* cylinders [8, 15], superquadrics, voxels [7]). We argue that a richer representation of shape is needed to make future strides in building better generative models. Discriminative methods [1, 2, 10, 13, 16, 17], more recently introduced specifically for the pose estimation task, do not address estimation of the body shape; in fact, they are specifically designed to be invariant to body shape variations. Any real-world system must be able to estimate both body shape and pose simultaneously.

Discriminative approaches to pose estimation attempt to learn a direct mapping from image features to 3D pose from either a single image [1, 14, 17] or multiple approximately calibrated views [9]. These approaches tend to use silhouettes [1, 9, 14] and sometimes edges [16, 17] as image features and learn a probabilistic mapping in the form of Nearest Neighbor (NN) search, regression [1], mixture of regressors [2], mixture of Baysian experts [17], or specialized mappings [14]. While effective and fast, they are inherently limited by the amount and the quality of the training data. More importantly they currently do not address estimation of the body shape itself. Body shape estimation (independent of the pose) has many applications in biometric authentication and consumer application domains.

Simplified models of body shape have a long history in computer vision and provide a relatively low dimensional description of the human form. More detailed triangulated mesh models obtained from laser range scans have been viewed as too high dimensional for vision applications. Moreover, mesh models of individuals lack a convenient, low-dimensional, parameterization to allow fitting to new subjects. In this paper we use the SCAPE model (Shape Completion and Animation of PEople) [3] which provides a low-dimensional parameterized mesh that is learned from a database of 3D range scans of different people. The SCAPE model captures correlated body shape deformations of the body due to the identity of the person and their non-rigid muscle deformation due to articulation. This model has been shown to allow tractable estimation of parameters from multi-view silhouette image features [5, 11] and from monocular images in scenes with point lights and cast shadows [4].

In [5] the SCAPE model is projected into multiple calibrated images and an iterative importance sampling method is used for inference of the pose and shape that best explain the observed silhouettes. Alternatively, in [11] visual hulls are constructed from many silhouette images and the Iterative Closest Point (ICP) algorithm is used to extract the pose by registering the volumetric features with SCAPE. Both [5] and [11], however, require manual initialization to bootstrap estimation. In this paper we substitute discriminative articulated pose and shape estimation in place of manual initialization. In doing so, we extend the current models for discriminative pose estimation to deal with the estimation of shape, and couple the discriminative and generative methods for more robust combined estimation. Few combined discriminative and generative pose estimation methods that exist [16], typically require temporal image data and do not address shape estimation problem.

For discriminative pose and shape estimation we use a Mixture of Experts model, with kernel linear regression as experts, to learn a direct probabilistic mapping between monocular silhouette contour features and the SCAPE parameters. To our knowledge this is the first work that has attempted to recover the 3D shape of the human body from monocular image directly. While the results are typically noisy, they are appropriate as initialization for the more precise generative refinement process. For generative optimization we make use of the method proposed in [5] where the silhouettes are predicted in multiple views given the pose and shape parameters of the SCAPE model and are compared to the observed silhouettes using a Chamfer distance measure. For training data we use the SCAPE model to generate pairs of 3D body shapes and projected image silhouettes. Evaluation is performed on sequences of two subjects performing free-style motion. We are able to predict pose, shape, and simple biometric measurements for the subjects from images captured by 4 synchronized cameras. We also show results for 3D shape estimation from monocular images.

The contributions of this paper are two fold: (1) we formulate a discriminative model for estimating the pose and shape directly from monocular image features, and (2) we couple this discriminative method with a generative stochastic optimization for detailed estimation of pose and the shape.

## 2   SCAPE Body Model

In this section we briefly introduce the SCAPE body model; for details the reader is referred to [3]. A low-dimensional mesh model is learned using principal component analysis applied to a registered database of range scans. The SCAPE model is defined by a set of parameterized deformations that are applied to a reference mesh that consists of $T$ triangles $\{\Delta x_t | t \in [1, ..., T]\}$ (here $T = 25,000$). Each of the triangles in the reference mesh is defined by three vertices in 3D space, $(v_{t,1}, v_{t,2}, v_{t,3})$, and has a corresponding associated body part index $p_t \in [1, ..., P]$ (we work with the model that has $P = 15$ body parts corresponding to torso, pelvis, head, and 3 segments for each of the upper and lower extremities). For convenience, the triangles of the mesh are parameterized by the edges, $\Delta x_t = (v_{t,2} - v_{t,1}, v_{t,3} - v_{t,1})$, instead of the vertices themselves. Estimating the shape and articulated pose of the body amounts to estimating parameters, $\mathbf{Y}$, of the deformations required to produce the mesh $\{\Delta y_t | t \in [1, ..., T]\}$, the projection of which matches the image evidence. The state-space of the model can be expressed by a vector $\mathbf{Y} = \{\tau, \theta, \nu\}$, where $\tau \in \mathbb{R}^3$ is the global 3D position for the body, $\theta \in \mathbb{R}^{37}$ is the joint-angle parameterization of the articulation with respect to the skeleton (encoded using Euler angles), and $\nu \in \mathbb{R}^9$ is the shape parameters encoding the identity-specific shape of the person. Given a set of estimated parameters $\mathbf{Y}$ a new mesh $\{\Delta y_t\}$ can be produced using:

$$\Delta y_t = R_{p_t}(\theta) S(\nu) Q(R_{p_t}(\theta)) \Delta x_t \tag{1}$$

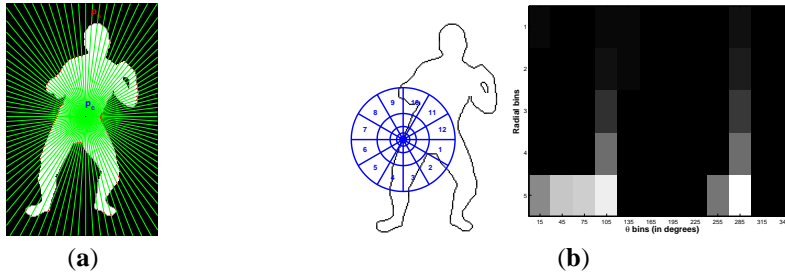

<div align="center">(a)           (b)</div>

Figure 1: **Silhouette contour descriptors.** Radial Distance Function (RDF) encoding of the silhouette contour is illustrated in (**a**); Shape Context (SC) encoding of a contour sample point in (**b**).

where $R_{p_t}(\theta)$ is the rigid $3 \times 3$ rotation matrix for a part $p_t$ and is a function of the joint angles $\theta$; $S(\nu)$ is the linear $3 \times 3$ transformation matrix modeling subject-specific shape variation as a function of the shape-space parameters $\nu$; $Q(R_{p_t}(\theta))$ is a $3 \times 3$ residual transformation corresponding to the non-rigid articulation-induced deformations (*e.g.* bulging of muscles). Notice, that $Q()$ is simply a learned linear function of the rigid rotation and has no independent parameters. To learn $Q()$ we minimize the residual in the least-squared sense between the set of 70 registered scans of one person under different (but known) articulations. It is also worth mentioning that body shape linear deformation sub-space, $S(\nu) = U_s\nu + \mu_s$, is learned from a set of 10 meshes of different people in full correspondence using PCA; hence $\nu$ can be interpreted as a vector of linear coefficients corresponding to eigen-directions of the shape-space that characterize a given body shape.

## 3 Features

In this work we make use of silhouette features for both discriminative and generative estimation of pose and shape. Silhouettes are commonly used for human pose estimation [1, 2, 13, 15, 17]; while limited in their representational power, they are easy to estimate from images and fast to synthesize from a mesh model. The framework introduced here, however, is general and can easily be extended to incorporate richer features (*e.g.* edges [15], dense region descriptors [16] such as SIFT or HOG, or hierarchical descriptors [10] like HMAX, Hyperfeatures, Spatial Pyramid). The use of such richer feature representations will likely improve both discriminative and generative estimation.

**Histograms of shape context.** Shape contexts (SC) [6] are rich descriptors based on the local shape-based histograms of the contour points sampled from the external boundary of the silhouette. At every sampled boundary point the shape context descriptor is parameterized by the number of orientation bins, $\phi$, number of radial-distance bins, $r$, and the minimum and maximum radial distances denoted by $r_{in}$ and $r_{out}$ respectively. As in [1] we achieve scale invariance by making $r_{out}$ a function of the overall silhouette height and normalizing the individual shape context histogram by the sum over all histogram bins. Assuming that $N$ contour points are chosen, at random, to encode the silhouette, the full feature vector can be represented using $\phi r N$ bin histogram. Even for moderate values of $N$ this produces high dimensional feature vectors that are hard to deal with.

To reduce the silhouette representation to a more manageable size, a secondary histogramming was introduced by Agarwal and Triggs in [1]. In this, *bag-of-words* style model, the shape context space is vector quantized into a set of $K$ clusters (*a.k.a.* codewords). The $K = 100$ center codebook is learned by running k-means clustering on the combined set of shape context vectors obtained from the large set of training silhouettes. Once the codebook is learned, the quantized $K$-dimensional histograms are obtained by voting into the histogram bins corresponding to codebook entries. Soft voting has been shown [1] to reduce effects of spatial quantization. The final descriptor $\mathbf{X}_{sc} \in \mathbb{R}^K$ is normalized to unit length, to ensure that silhouettes that contain different number of contour points can be compared.

The resulting codebook shape context representation is translation and scale invariant by definition. Following the prior work [1, 13] we let $\phi = 12$, $r = 5$, $r_{in} = 3$, and $r_{out} = \kappa h$ where $h$ is the height of the silhouette and $\kappa$ is typically $\frac{1}{4}$ ensuring integration of contour points over regions roughly similar to the limb size [1]. For shape estimation, we found that combining features across multiple spatial scales (*e.g.* $\kappa = \{\frac{1}{4}, \frac{1}{2}, ...\}$) to be more effective.

**Radial distance function.** The Radial Distance Function (RDF) features are defined by a feature vector $\mathbf{X}_{rdf} = \{p_c, ||p_1 - p_c||, ||p_2 - p_c||, ..., ||p_N - p_c||\}$, where $p_c \in \mathbb{R}^2$ is the centroid of the image silhouette, and $p_i$ is the point on the silhouette outer contour; hence $||p_i - p_c|| \in \mathbb{R}$ measures the maximal object extent in the particular direction denoted by $i$ from the centroid. For all experiments, we use $N = 100$ points, resulting in the $\mathbf{X}_{rdf} \in \mathbb{R}^{102}$. We explicitly ensure that the dimensionality of the RDF descriptor is comparable to that of shape context introduced above. Unlike the shape context descriptor, the RDF feature vector is neither scale nor translation invariant. Hence, RDF features are only suited for applications where camera calibration is known and fixed.

## 4  Discriminative estimation of pose and shape

To produce initial estimates for the body pose and/or shape in 3D from image features, we need to model the conditional distribution $p(\mathbf{Y}|\mathbf{X})$ of the 3D body state $\mathbf{Y}$ given the set of 2D features $\mathbf{X}$. Intuitively this conditional mapping should be related to the inverse of the camera projection matrix and, as with many inverse problems, is highly ambiguous. To model this non-linear relationship we use a Mixtures of Experts (MoE) model to represent the conditionals [2, 17].

The parameters of the MoE model are learned by maximizing the log-likelihood of the training data set $\mathcal{D} = \{(x^{(1)}, y^{(1)}), ..., (x^{(N)}, y^{(N)})\}$ consisting of $N$ input-output pairs $(x^{(i)}, y^{(i)})$. We use an iterative Expectation Maximization (EM) algorithm, based on type-II maximum likelihood, to learn parameters of the MoE. Our model for the conditional can be written as:

$$p(\mathbf{Y}|\mathbf{X}) \propto \sum_{k=1}^{M} p_{e,k}(\mathbf{Y}|\mathbf{X}, \Theta_{e,k}) p_{g,k}(k|\mathbf{X}, \Theta_{g,k}) \qquad (2)$$

where $p_{e,k}$ is the probability of choosing pose $\mathbf{Y}$ given the input $\mathbf{X}$ according to the $k$-th expert, and $p_{g,k}$ is the probability of that input being assigned to the $k$-th expert using an input sensitive gating network; in both cases $\Theta$ represents the parameters of the mixture and gate distributions respectively.

For simplicity and to reduce complexity of the experts we choose kernel linear regression with constant offset, $\mathbf{Y} = \beta\mathbf{X} + \alpha$, as our expert model, which allows us to solve for the parameters $\Theta_{e,k} = \{\beta_k, \alpha_k, \Lambda_k\}$ analytically using the weighted linear regression, where $p_{e,k}(\mathbf{Y}|\mathbf{X}, \Theta_{e,k}) = \frac{1}{\sqrt{(2\pi)^n|\Lambda_k|}} \exp^{-\frac{1}{2}\Delta_k^T \Lambda_k^{-1} \Delta_k}$, and $\Delta_k = \mathbf{Y} - \beta_k\mathbf{X} - \alpha_k$.

Pose estimation is a high dimensional and ill-conditioned problem, so simple least squares estimation of the linear regression matrix parameters typically produces severe over-fitting and poor generalization. To reduce this, we add smoothness constraints on the learned mapping. We use a damped regularization term $R(\beta) = \lambda||\beta||^2$ that penalizes large values in the coefficient matrix $\beta$, where $\lambda$ is a regularization parameter. Larger values of $\lambda$ will result in overdamping, where the solution will be underestimated, small values of $\lambda$ will result in overfitting and possibly ill-conditioning. Since the solution of the ridge regressors is not symmetric under the scaling of the inputs, we normalize the inputs $\{x^{(1)}, x^{(2)}, ..., x^{(N)}\}$ by the standard deviation in each dimension respectively before solving.

Weighted ridge regression solution for the parameters $\beta_k$ and $\alpha_k$ can be written in matrix notation as follows,

$$\begin{bmatrix} \beta_k \\ \alpha_k \end{bmatrix}^T = \begin{bmatrix} \mathcal{D}_{\mathbf{X}}^T \operatorname{diag}(Z_k)\, \mathcal{D}_{\mathbf{X}} + \operatorname{diag}(\lambda) & Z_k \\ Z_k^T & Z_k^T Z_k \end{bmatrix}^{-1} \begin{bmatrix} \mathcal{D}_{\mathbf{X}}^T \\ Z_k^T \end{bmatrix} \operatorname{diag}(Z_k)\, \mathcal{D}_{\mathbf{Y}}, \qquad (3)$$

where $Z_k = [z_k^{(1)}, z_k^{(2)}, ..., z_k^{(N)}]^T$ is the vector of ownership weights described later in the section and $\operatorname{diag}(Z_k)$ is diagonal matrix with $Z_k$ on the diagonal; $\mathcal{D}_{\mathbf{X}} = [x^{(1)}, x^{(2)}, ..., x^{(N)}]$ and $\mathcal{D}_{\mathbf{Y}} = [y^{(1)}, y^{(2)}, ..., y^{(N)}]$ are vectors of inputs and outputs from the training data $\mathcal{D}$.

Maximization for the gate parameters can be done analytically as well. Given the gate model, $p_{g,k}(k|\mathbf{X}, \Theta_{g,k}) = \frac{1}{\sqrt{(2\pi)^n|\Sigma_k|}} \exp^{-\frac{1}{2}(\mathbf{X}-\mu_k)^T \Sigma_k^{-1}(\mathbf{X}-\mu_k)}$ maximization of the gate parameters $\Theta_{g,k} = \{\Sigma_k, \mu_k\}$ becomes similar to the mixture of Gaussians estimation, where $\mu_k = \sum_{n=1}^{N} z_k^{(n)} x^{(n)} / \sum_{n=1}^{N} z_k^{(n)}$, $\Sigma_k = \frac{1}{\sum_{n=1}^{N} z_k^{(n)}} \sum_{n=1}^{N} z_k^{(n)} [x^{(n)} - \mu_k][x^{(n)} - \mu_k]^T$ and $z_k^n$ is the

estimated ownership weight of the example $n$ by the expert $k$ estimated by expectation

$$z_k^{(n)} = \frac{p_{e,k}(y^{(n)}|x^{(n)}, \Theta_{e,k})p_{g,k}(k|x^{(n)}, \Theta_{g,k})}{\sum_{j=1}^{M} p_{e,j}(y^{(n)}|x^{(n)}, \Theta_{e,j})p_{g,j}(j|x^{(n)}, \Theta_{g,j})}. \tag{4}$$

The above outlines the full EM procedure for the MoE model. We learn three separate models for shape, $p(\nu|\mathbf{X})$, articulated pose, $p(\theta|\mathbf{X})$ and global position, $p(\tau|\mathbf{X})$. Similar to [2] we initialize the EM learning by clustering the output 3D poses using the K-means procedure.

**Implementation details.** For articulated pose and shape we experimented with using both RDF and SC features (global position requires RDF features since SC is location and scale invariant). SC features tend to work better for pose estimation where as RDF features perform better for shape estimation. Hence, we learn $p(\nu|\mathbf{X}_{rdf})$, $p(\theta|\mathbf{X}_{sc})$ and $p(\tau|\mathbf{X}_{rdf})$. In cases where calibration is unavailable, we estimate the shape using $p(\nu|\mathbf{X}_{sc})$ which tends to produce reasonable results but cannot estimate the overall height. We estimate the number of mixture components, $M$, and regularization parameter, $\lambda$, by learning a number of models and cross validating on the withheld dataset.

## 5 Generative stochastic optimization of pose and shape

Generative stochastic state estimation, as in [5], is handled within an iterative importance sampling framework [8]. To this end, we represent the posterior distribution over the state (that includes both pose and shape), $p(\mathbf{Y}|I) \propto p(I|\mathbf{Y})p(\mathbf{Y})$, using a set of $N$ weighted samples $\{y_i, \pi_i\}_{i=1}^{N}$, where $y_i \sim q(\mathbf{Y})$ is a sample drawn from the importance function $q(\mathbf{Y})$ and $\pi_i \propto \frac{p(I|y_i)p(y_i)}{q(y_i)}$ is an associated normalized weight. As in [5] we make no rigorous probabilistic claims about the generative model, but rather use it as effective means of performing stochastic search. As required by the annealing framework, we define a set of importance functions $q_k(\mathbf{Y})$ from which we draw samples at each respective iteration $k$. We define importance functions recursively using a smoothed version of posterior from the previous iteration $q_{k+1}(\mathbf{Y}) = \sum_{i=1}^{N} \pi_i^{(k)} \mathcal{N}(y_i^{(k)}, \Sigma^{(k)})$, encoded using a kernel Gaussian density with iteration dependent bandwidth parameter $\Sigma^{(k)}$. To avoid effects of local optima, the likelihood is annealed as follows: $p_k(I|\mathbf{Y}) = [p(I|\mathbf{Y})]^{T_k}$ at every iteration, where $T_k$ is the temperature parameter. As a result, effects of peaks in the likelihood are introduced slowly.

To initiate the stochastic search an initial distribution is needed. The high dimensionality of the state space requires this initial distribution to be relatively close to the solution in order to reach convergence. Here we make use of the discriminative pose and shape estimate from Section 4 to give us the initial distribution for the posterior. In particular, given the discriminative model for the shape, $p(\nu|\mathbf{X})$, position, $p(\tau|\mathbf{X})$, and articulated pose, $p(\theta|\mathbf{X})$, of the body, we can let (with slight abuse of notation) $y_i^{(0)} \sim [p(\tau|\mathbf{X}), p(\theta|\mathbf{X}), p(\nu|\mathbf{X})]$ and $\pi_i^{(0)} = 1/N$ for $i \in [1, ..., N]$.

The outlined stochastic optimization framework also requires an image likelihood function, $p(I|\mathbf{Y})$, that measures how well our model under a given state $\mathbf{Y}$ matches the image evidence, $I$, obtained from one or multiple synchronized cameras. We adopt the likelihood function introduced in [5] that measures the similarity between observed and hypothesized silhouettes. For a given camera view, a foreground silhouette is computed using a shadow-suppressing background subtraction procedure and is compared to the silhouette obtained by projecting the SCAPE model subject to the hypothesized state into the image plane (given calibration parameters of the camera). Pixels in the non-overlapping regions are penalized by the distance to the closest contour point of the silhouette. This is made efficient by the use of Chamfer distance map precomputed for both silhouettes.

## 6 Experiments

**Datasets.** In this paper we make use of 3 different datasets. The **training dataset**, used to learn discriminative MoE models and codeword dictionary for SC, was generated by synthesizing 3000 silhouette images obtained by projecting corresponding SCAPE body models into an image plane using calibration parameters of the camera. SCAPE body models, in turn, were generated by randomly sampling the pose from a database of motion capture data (consisting of generally non-cyclic random motions) and the body shape coefficient from a uniform distribution centered at the mean shape. Similar synthetic **test dataset** was constructed consisting of 597 silhouette-SCAPE body

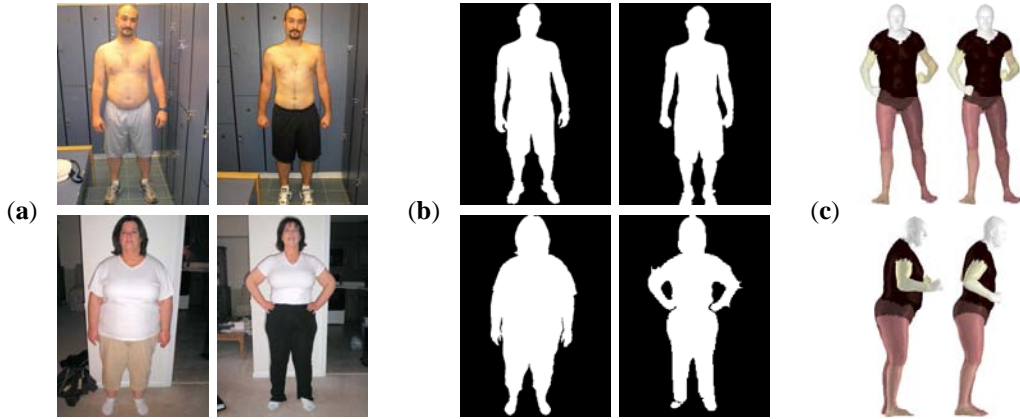

Figure 2: **Discriminative estimation of weight loss.** Two images of a subject *before* and *after* weight loss are shown in (**a**) on the left and right respectively. The images were downloaded from the web (Google) and manually segmented (**b**). The estimated shape and pose obtained by our discriminative estimation procedure is shown in (**c**). In bottom row, we manually rotated the model $90\ degrees$ for better visibility of the shape variation. Since camera calibration is unavailable, we use $p(\nu|\mathbf{X}_{sc})$ and normalize the *before* and *after* shapes to the same reference height. Our method estimated that the person illustrated in the top row lost $22\ lb$ and the one illustrated in the bottom row $- 32\ lb$; web-reported weight loss for the two subjects was $24\ lb$ and $64\ lb$ respectively. Notice that the neutral posture assumed in images was not present in our training data set, causing visible artifacts with estimation of the arm pose. Also, the bottom example pushes the limits of our current shape model which was trained using only $10$ scans of people, none close to the desired body shape.

model pairs. In addition, we collected a **real dataset** consisting of hardware-synchronized motion capture and video collected using $4$ cameras. Two subjects were captured performing roughly the same class of motions as in the training dataset.

**Discriminative estimation of shape.** Results of using the MoE model, similar to the one introduced here, for pose estimation have previously been reported in [2] and [17]. Our experience with the articulated pose estimation was similar and we omit supporting experiments due to lack of space. For discriminative estimation of shape we quantitatively compared SC and RDF features, by training two MoE models $p(\nu|\mathbf{X}_{sc})$ and $p(\nu|\mathbf{X}_{rdf})$, and found the latter to perform better when camera calibration is available (on the average we achieve a 19.3 % performance increase over simply using the mean shape). We attribute the superior performance of RDF features to their sensitivity to the silhouette position and scale, that allows for better estimation of overall height of the body.

Given the shape we can also estimate the volume of the body and assuming constant density of water, compute the weight of the person. To illustrate this we estimate approximate weight loss of a person from monocular uncalibrated images (see Figure 2). Please note that this application is a proof of concept and not a rigorous experiment[1]. In principle, the SCAPE model is not ideal for weight calculations, since non-rigid deformations caused by articulations of the body will result in (unnatural) variations in weight. In practice, however, we found such variations produce relatively minor artifacts. The weight calculations are, on the other hand, very sensitive to the body shape.

**Combining discriminative and generative estimation.** Lastly we tested the performance of the combined discriminative and generative framework by estimating articulated pose, shape and biometric measurements for people in our **real dataset**. Results of biometric measurement estimates can be seen in Figure 3; corresponding visual illustration of results is shown in Figure 4.

**Analysis of errors.** Rarely our system does produce poor pose and/or shape estimates. Typically these cases can be classified into two categories: (1) minor errors that only effect the pose and are artifacts of local optima or (2) more significant errors that effect the shape and result from poor initial distribution over the state produced by the discriminative method. The latter arise as a result of 180–degree view ambiguity and/or pose configuration ambiguities, due to symmetry, in the silhouettes.

|  | Biometric Feature | Actual | Discriminative | | Disc. + Generative | | GT + Generative | |
|---|---|---|---|---|---|---|---|---|
|  |  |  | Mean | Std | Mean | Std | Mean | Std |
| A (34) | Height ($mm$) | 1780 | 1716.1 | 41.9 | 1776.2 | 43.8 | 1796.9 | 22.9 |
|  | Arm Span ($mm$) | 1597 | 1553.6 | 39.7 | 1597.3 | 58.0 | 1607.7 | 30.7 |
|  | Weight ($kg$) | 88 | 83.62 | 8.94 | 83.37 | 8.01 | 85.83 | 3.73 |
| B (30) | Height ($mm$) | 1825 | 1703.8 | 88.8 | 1751.0 | 95.2 | 1844.1 | 63.8 |
|  | Arm Span ($mm$) | 1668 | 1537.7 | 69.2 | 1547.5 | 91.4 | 1659.0 | 29.1 |
|  | Weight ($kg$) | 63 | 80.63 | 18.53 | 64.98 | 9.27 | 66.33 | 4.69 |

Figure 3: **Estimating basic biometric measurements.** Figure illustrates basic biometric measurements (height, arm span[3] and weight) recovered for two subjects A and B. Mean and standard deviation reported over 34 and 30 frames for subject A and B respectively. Every 25-th frame from two sequence obtained using 4 synchronized cameras was chosen for estimation. The actual measured values for the two subjects are shown in the left column. Estimates obtained using discriminative only and discriminative followed by generative shape estimation methods are reported in the next two columns. Discriminative method used only one view for estimation, where as generative method used all 4 views to obtain a better fit. Last column reports estimates obtained using ground truth pose and mean shape as initialization for the generative fit (this is the algorithm proposed in [5]). Notice that generative estimation significantly refines the discriminative estimates. In addition, our approach, that unlike [5] does not require manual initialization, performs comparably (and sometimes marginally better than [5]) in terms of mean performance (but has roughly twice the variance).

## 7  Discussion and Conclusions

We have presented a method for automatic estimation of articulated pose and shape of people from images. Our approach goes beyond prior work in that it is able to estimate a detailed parametric model (SCAPE) directly from images without requiring manual intervention or initialization. We found that the discriminative model produced an effective initialization for generative optimization procedure and that biometric measurements from the recovered shape were comparable to those produced by prior approaches that required manual initialization [5]. We also introduced and addressed the problem of discriminative estimation of shape from monocular calibrated and un-calibrated images. More accurate shape estimates from monocular data will require richer image descriptors.

A number of straightforward extensions to our model will likely yeld immediate improvement in performance. Among such, is the use of temporal consistency in the discriminative pose (and perhaps shape) estimation [17] and dense image descriptors [10]. In addition, in this work we estimated the shape space of the SCAPE model from only 10 body scans, as a result the learned shape space is rather limited in its expressive power. We belive some of the artifacts of this can be observed in Figure 2 where the weight of the heavier woman is underestimated.

**Acknowledgments.** This work was supported by NSF grants IIS-0534858 and IIS-0535075 and a gift from Intel Corp. We also thank James Davis and Dragomir Anguelov for discussions and data.

## Footnotes

[1]The "ground truth" weight change here is self reported and gathered from the Internet.

[3] Arm span is defined as the distance between knuckles of left and right arm fully extended in 'T'-pose [5].

## References

[1] A. Agarwal and B. Triggs. Recovering 3D human pose from monocular images, *IEEE Transactions on Pattern Analysis and Machine Intelligence*, Vol. 28, No. 1, pp. 44–58, 2006.

[2] A. Agarwal and B. Triggs. Monocular human motion capture with a mixture of regressors, *IEEE Workshop on Vision for Human-Computer Interaction*, 2005.

[3] D. Anguelov, P. Srinivasan, D. Koller, S. Thrun, J. Rodgers and J.Davis. SCAPE: Shape Completion and Animation of PEople, *ACM Transactions on Graphics (SIGGRAPH)*, Vol. 24(3), pp. 408–416, 2005.

[4] A. Balan, M. J. Black, H. Haussecker and L. Sigal. Shining a light on human pose: On shadows, shading and the estimation of pose and shape, *International Conference on Computer Vision (ICCV)*, 2007.

[5] A. Balan, L. Sigal, M. Black, J. Davis and H. Haussecker. Detailed human shape and pose from images, *IEEE Conference on Computer Vision and Pattern Recognition (CVPR)*, 2007.

[6] S. Belongie, J. Malik and J. Puzicha. Matching shapes, *ICCV*, pp. 454–461, 2001.

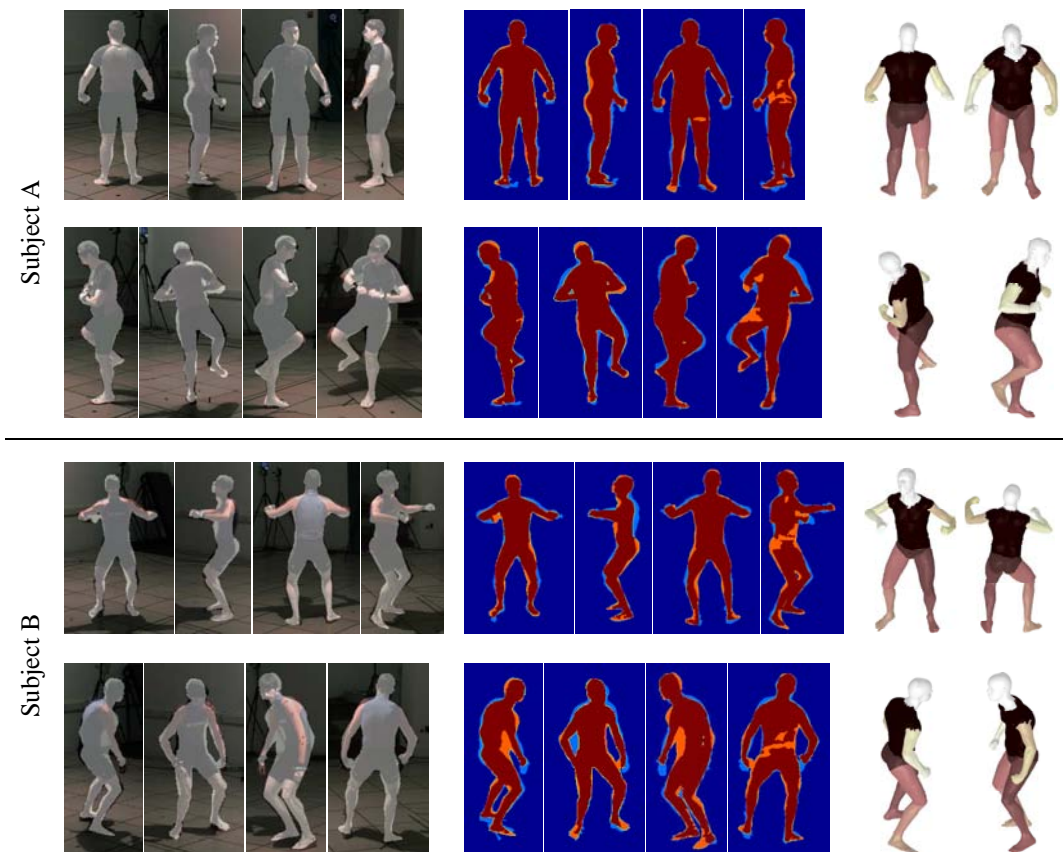

Figure 4: **Visualizing pose and shape estimation.** Examples of simultaneous pose and shape estimation for subjects A and B are shown on top and bottom respectively. Results are obtained by discriminatively estimating the distribution over the initial state and then refining this distribution via generative local stochastic search. Left column illustrates projection of the estimated model into all 4 views. Middle column shows the projection of the model onto image silhouettes, where light blue denotes image silhouette, dark red projection of the model and orange non-silhouette regions that overlap with the projection. On the right are the two views of the estimated 3D model.

[7] K. M. Cheung, S. Baker and T. Kanade. Shape-from-silhouette of articulated objects and its use for human body kinematics estimation and motion capture, *CVPR*, Vol. 1, pp. 77–84, 2003.

[8] J. Deutscher, A. Blake and I. Reid. Articulated body motion capture by annealed particle filtering, *IEEE Conference on Computer Vision and Pattern Recognition (CVPR)*, Vol. 2, pp. 126–133, 2000.

[9] K. Grauman, G. Shakhnarovich, T. Darrell. Inferring 3D structure with a statistical image-based shape model, *IEEE International Conference on Computer Vision (ICCV)*, pp. 641–648, 2003.

[10] A. Kanaujia, C. Sminchisescu and D. Metaxas. Semi-supervised Hierarchical Models for 3D Human Pose Reconstruction, *IEEE Conference on Computer Vision and Pattern Recognition (CVPR)*, 2007.

[11] L. Muendermann, S. Corazza and T. Andriacchi. Accurately measuring human movement using articulated ICP with soft-joint constraints and a repository of articulated models, *CVPR*, 2007.

[12] R. Plankers and P. Fua. Articulated soft objects for video-based body modeling, *ICCV*, 2001.

[13] R. W. Poppe and M. Poel. Comparison of silhouette shape descriptors for example-based human pose recovery, *IEEE Conference on Automatic Face and Gesture Recognition (FG 2006)*, pp. 541–546, 2006.

[14] R. Rosales and S. Sclaroff. Learning Body Pose Via Specialized Maps, *NIPS*, 2002.

[15] L. Sigal, S. Bhatia, S. Roth, M. J. Black and M. Isard Tracking Loose-limbed People, *IEEE Conference on Computer Vision and Pattern Recognition (CVPR)*, Vol. 1, pp. 421–428, 2004.

[16] C. Sminchisescu, A. Kanajujia and D. Metaxas. Learning Joint Top-Down and Bottom-up Processes for 3D Visual Inference, *CVPR*, Vol. 2, pp. 1743–1752, 2006.

[17] C. Sminchisescu, A. Kanaujia, Z. Li and D. Metaxas. Discriminative density propagation for 3D human motion estimation, *CVPR*, Vol. 1, pp. 390–397, 2005.

